# Modelling Seasonality and Trends in Daily Rainfall Data

**Peter M Williams**
School of Cognitive and Computing Sciences
University of Sussex
Falmer, Brighton BN1 9QH, UK
email: peterw@cogs.susx.ac.uk

## Abstract

This paper presents a new approach to the problem of modelling daily rainfall using neural networks. We first model the conditional distributions of rainfall amounts, in such a way that the model itself determines the order of the process, and the time-dependent shape and scale of the conditional distributions. After integrating over particular weather patterns, we are able to extract seasonal variations and long-term trends.

## 1 Introduction

Analysis of rainfall data is important for many agricultural, ecological and engineering activities. Design of irrigation and drainage systems, for instance, needs to take account not only of mean expected rainfall, but also of rainfall volatility. In agricultural planning, changes in the annual cycle, e.g. advances in the onset of winter rain, are significant in determining the optimum time for planting crops. Estimates of crop yields also depend on the distribution of rainfall during the growing season, as well as on the overall amount. Such problems require the extrapolation of longer term trends as well as the provision of short or medium term forecasts.

## 2 Occurrence and amount processes

Models of daily precipitation commonly distinguish between the **occurrence** process, i.e. whether or not it rains, and the **amount** process, i.e. how much it rains, if it does. The occurrence process is often modelled as a two-state Markov chain of first or higher order. In discussion of [12], Katz traces this approach back to Quetelet in 1852. A first order chain has been considered adequate for some weather stations, but second or higher order models may be required for others, or at different times of year. Non-stationary Markov chains have been used by a number of investigators, and several approaches have been taken

to the problem of seasonal variation, e.g. using Fourier series to model daily variation of parameters [16, 12, 15].

The amount of rain $X$ on a given day, assuming it rains, normally has a roughly exponential distribution. Smaller amounts of rain are generally more likely than larger amounts. Several models have been used for the amount process. Katz & Parlange [9], for example, assume that $\sqrt[n]{X}$ has a normal distribution, where $n$ is a positive integer empirically chosen to minimise the skewness of the resulting historical distribution. But use has more commonly been made of a gamma distribution [7, 8, 12] or a mixture of two exponentials [16, 15].

## 3    Stochastic model

The present approach is to deal with the occurrence and amount processes jointly, by assuming that the distribution of the amount of rain on a given day is a mixture of a discrete and continuous component. The discrete component relates to rainfall occurrence and the continuous component relates to rainfall amount on rainy days.

We use a gamma distribution for the continuous component.[1] This has density proportional to $x^{\nu-1}e^{-x}$ to within an adjustable scaling of the $x$-axis. The shape parameter $\nu > 0$ controls the ratio of standard deviation to mean. It also determines the location of the mode, which is strictly positive if $\nu > 1$. For certain patterns of past precipitation, larger amounts may be more likely on the following day than smaller amounts. Specifically the distribution of the amount $X$ of rain on a given day is modelled by the three parameter family

$$P(X > x) = \begin{cases} 1 & \text{if } x < 0 \\ \alpha \Gamma\left(\nu, \dfrac{x}{\theta}\right) & \text{if } x \geq 0 \end{cases} \tag{1}$$

where $0 \leq \alpha \leq 1$ and $\nu, \theta > 0$ and

$$\Gamma(\nu, z) = \Gamma(\nu)^{-1} \int_z^\infty y^{\nu-1} e^{-y} \, dy$$

is the incomplete gamma function. For $\alpha < 1$, there is a discontinuity at $x = 0$ corresponding to the discrete component. Putting $x = 0$, it is seen that $\alpha = P(X > 0)$ is the probability of rain on the day in question. The mean daily rainfall amount is $\alpha\nu\theta$ and the variance is $\alpha\nu\{1 + \nu(1 - \alpha)\}\theta^2$.

## 4    Modelling time dependency

The parameters $\alpha, \nu, \theta$ determining the conditional distribution for a given day, are understood to depend on the preceding pattern of precipitation, the time of year etc. To model this dependency we use a neural network with inputs corresponding to the conditioning events, and three outputs corresponding to the distributional parameters.[2] Referring to the activations of the three output units as $z^\alpha$, $z^\nu$ and $z^\theta$, we relate these to the distributional parameters by

$$\alpha = \frac{1}{1 + \exp z^\alpha} \qquad \nu = \exp z^\nu \qquad \theta = \exp z^\theta \tag{2}$$

in order to ensure an unconstrained parametrization with $0 < \alpha < 1$ and $\nu, \theta > 0$ for any real values of $z^\alpha, z^\nu, z^\theta$.

On the input side, we first need to make additional assumptions about the statistical properties of the process. Specifically it is assumed that the present is stochastically independent of the distant past in the sense that

$$P(X_t > x \mid X_{t-1}, \ldots, X_0) = P(X_t > x \mid X_{t-1}, \ldots, X_{t-T}) \qquad (t > T) \qquad (3)$$

for a sufficiently large number of days $T$. In fact the stronger assumption will be made that

$$P(X_t > x \mid X_{t-1}, \ldots, X_0) = P(X_t > x \mid R_{t-1}, \ldots, R_{t-T}) \qquad (t > T) \qquad (4)$$

where $R_t = (X_t > 0)$ is the event of rain on day $t$. This assumes that today's rainfall amount depends stochastically only on the occurrence or non-occurrence of rain in the recent past, and not on the actual amounts. Such a simplification is in line with previous approaches [8, 16, 12]. For the present study $T$ was taken to be 10.

To assist in modelling seasonal variations, cyclic variables $\sin \tau$ and $\cos \tau$ were also provided as inputs, where $\tau = 2\pi t/D$ and $D = 365.2422$ is the length of the tropical year. This corresponds to using Fourier series to model seasonality [16, 12] but with the number of harmonics adaptively determined by the model.[3] To allow for non-periodic non-stationarity, the current value of $t$ was also provided as input.

## 5   Model fitting

Suppose we are given a sequence of daily rainfall data of length $N$. Equation (4) implies that the likelihood of the full data sequence $(x_{N-1}, \ldots, x_0)$ factorises as

$$p(x_{N-1}, \ldots, x_0; \mathbf{w}) = p(x_{T-1}, \ldots, x_0) \prod_{t=T}^{N-1} p(x_t \mid r_{t-1}, \ldots, r_{t-T}; \mathbf{w}) \qquad (5)$$

where the likelihood $p(x_{T-1}, \ldots, x_0)$ of the initial sequence is not modelled and can be considered as a constant (compare [14]). Our interest is in the likelihood (5) of the actual sequence of observations, which is understood to depend on the variable weights $\mathbf{w}$ of the neural network. Note that $p(x_t \mid r_{t-1}, \ldots, r_{t-T}; \mathbf{w})$ is computed by means of the neural network outputs $z_t^\alpha, z_t^\nu, z_t^\theta$, using weights $\mathbf{w}$ and the inputs corresponding to time $t$.

The log likelihood of the data can therefore be written, to within a constant, as

$$\log p(x_{N-1}, \ldots, x_0; \mathbf{w}) = \sum_{t=T}^{N-1} \log p(x_t \mid r_{t-1}, \ldots, r_{t-T}; \mathbf{w})$$

or, more simply,

$$L(\mathbf{w}) = \sum_{t=T}^{N-1} L_t(\mathbf{w}) \qquad (6)$$

where from (1)

$$L_t(\mathbf{w}) = \begin{cases} \log(1 - \alpha_t) & \text{if } x_t = 0 \\ \log \alpha_t + (\nu_t - 1) \log x_t - \nu_t \log \theta_t - \log \Gamma(\nu_t) - x_t/\theta_t & \text{if } x_t > 0 \end{cases} \qquad (7)$$

where dependence of $\alpha_t, \nu_t, \theta_t$ on $\mathbf{w}$, and also on the data, is implicit.

To fit the model, it is useful to know the gradient $\nabla L(\mathbf{w})$. This can be computed using backpropagation if we know the partial derivatives of $L(\mathbf{w})$ with respect to network outputs. In view of (6) we can concentrate on a single observation and perform a summation.

Omitting subscript references to $t$ for simplicity, and recalling the links between network outputs and distributional parameters given by (2), we have

$$
\frac{\partial L}{\partial z^\alpha} = \begin{cases} -\alpha & \text{if } x = 0 \\ 1-\alpha & \text{if } x > 0 \end{cases}
$$

$$
\frac{\partial L}{\partial z^\nu} = \begin{cases} 0 & \text{if } x = 0 \\ \nu\psi(\nu) - \nu\log\frac{x}{\theta} & \text{if } x > 0 \end{cases}
\tag{8}
$$

$$
\frac{\partial L}{\partial z^\theta} = \begin{cases} 0 & \text{if } x = 0 \\ \nu - \frac{x}{\theta} & \text{if } x > 0 \end{cases}
$$

where

$$
\psi(\nu) = \frac{d}{d\nu}\log\Gamma(\nu) = \frac{\Gamma'(\nu)}{\Gamma(\nu)}
$$

is the digamma function of $\nu$. Efficient algorithms for computing $\log\Gamma(\nu)$ in (7) and $\psi(\nu)$ in (8) can be found in Press et al. [11] and Amos [1].

## 6 Regularization

Since neural networks are universal approximators, some form of regularization is needed. As in all statistical modelling, it is important to strike the right balance between jumping to conclusions (overfitting) and refusing to learn from experience (underfitting). For this purpose, each model was fitted using the techniques of [13] which automatically adapt the complexity of the model to the information content of the data, though other comparable techniques might be used. The natural interpretation of the regularizer is as a Bayesian prior. The Bayesian analysis is completed by integration over weight space. In the present case, this was achieved by fitting several models and taking a suitable mixture as the solution. On account of the large datasets used, however, the results are not particularly sensitive to this aspect of the modelling process.

## 7 Results for conditional distributions

The process was applied to daily rainfall data from 5 stations in south east England and 5 stations in central Italy.[4] The data covered approximately 40 years providing some 15,000 observations for each station. A simple fully connected network was used with a single layer of 13 input units, 20 hidden units and 3 output units corresponding to the 3 parameters of the conditional distribution shown in (2). As a consequence of the pruning features of the regularizer, the models described here used an average of roughly 65 of the 343 available parameters.

To illustrate the general nature of the results, Figure 1 shows an example from the analysis of an early part of the Falmer series. It is worth observing the succession of 16 rainy days from day 39 to day 54. The lefthand figure shows that the conditional probability of rain increases rapidly at first, and then levels out after about 5–7 days.[5] Similar behaviour is observed for successive dry days, for example between days 13 and 23. This suggests that the choice of 10 time lags was sufficient. Previous studies have used mainly first or second order Markov chains [16, 12]. Figure 1 confirms that conditional dependence

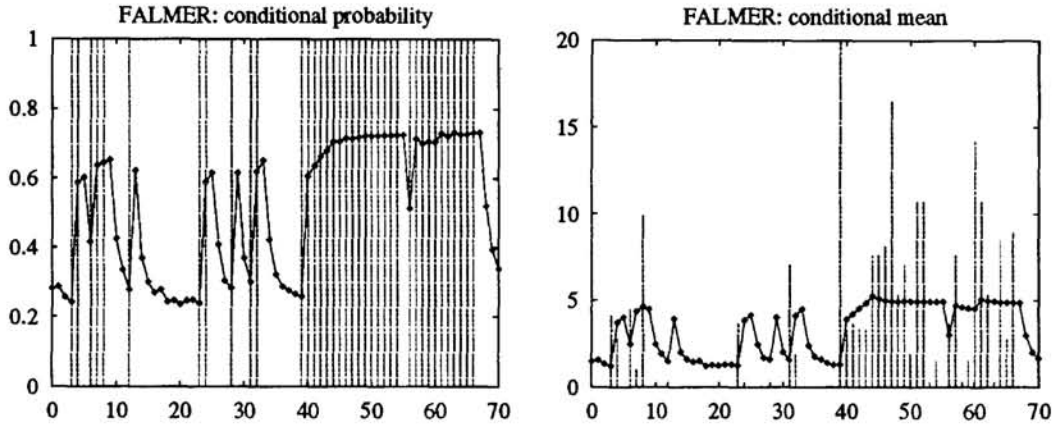

Figure 1: Results for the 10 weeks from 18 September to 27 November, 1951. The lefthand figure shows the conditional probability of rain for each day, with days on which rain occurred indicated by vertical lines. The righthand figure shows the conditional expected amount of rain in millimeters for the same period, together with the actual amount recorded.

decays rapidly at this station, at this time of year, but also indicates that it can persist for up to at least 5 days (compare [5, 4]).

## 8   Seasonality and trends

Conditional probabilities and expectations displayed in Figure 1 show considerable noise since they are realisations of random variables depending on the rainfall pattern for the last 10 days. For the purpose of analysing seasonal effects and longer term trends, it is more indicative to integrate out the noise resulting from individual weather patterns as follows.

Let $R_t$ denote the event $(X_t > 0)$ and let $\overline{R}_t$ denote the complementary event $(X_t = 0)$. The expected value of $X_t$ can then be expressed as

$$E(X_t) = \sum E(X_t \mid A_{t-1}, \ldots, A_{t-T}) \, P(A_{t-1}, \ldots, A_{t-T}) \qquad (9)$$

where each event $A_t$ stands for either $R_t$ or $\overline{R}_t$, and summation is over the $2^T$ possible combinations. Equation (9) takes the full modelled joint distribution over the variables $X_{N-1}, \ldots, X_0$ and extracts the marginal distribution for $X_t$. This should be distinguished from an unconditional distribution which might be estimated by pooling the data over all 40 years. $E(X_t)$ relates to a specific day $t$. Note that (9) also holds if $X_t$ is replaced by any integrable function of $X_t$, in particular by the indicator function of the event $(X_t > 0)$ in which case (9) expresses the probability of rain on that day.

Examining (9) we see that the conditional expectations in the first term on the right are known from the model, which supplies a conditional distribution not only for the sequence of events which actually occurred, but for any possible sequence over the previous $T$ days. It therefore only remains to calculate the probabilities $P(A_{t-1}, \ldots, A_{t-T})$ of $T$-day sequences preceding a given day $t$. Note that these are again time-dependent marginal probabilities, which can be calculated recursively from

$$
\begin{aligned}
P(A_t, \ldots, A_{t-T+1}) = {} & \\
& P(A_t \mid A_{t-1}, \ldots, A_{t-T+1} R_{t-T}) \, P(A_{t-1}, \ldots, A_{t-T+1} R_{t-T}) + \\
& P(A_t \mid A_{t-1}, \ldots, A_{t-T+1} \overline{R}_{t-T}) \, P(A_{t-1}, \ldots, A_{t-T+1} \overline{R}_{t-T})
\end{aligned}
$$

provided we assume a prior distribution over the $2^T$ initial sequences $(A_{T-1}, \ldots, A_0)$ as a base for the recursion. The conditional probabilities on the right are given by the model,

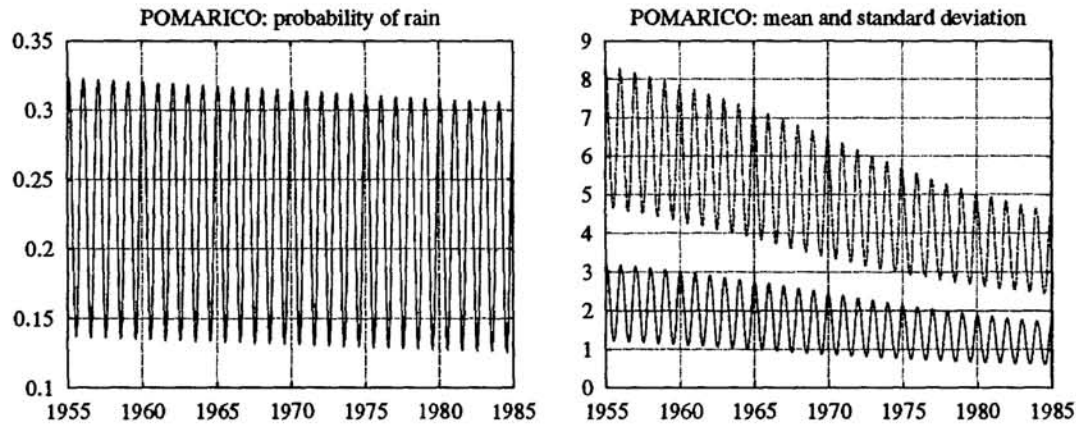

Figure 2: Integrated results for Pomarico from 1955–1985. The lefthand figure shows the daily probability of rain, indicating seasonal variation from a summer minimum to a winter maximum. The righthand figure shows the daily mean (above) and standard deviation (below) of rainfall amount in millimeters.

as before, and the unconditional probabilities are given by the recursion. It turns out that results are insensitive to the choice of initial distribution after about 50 iterations, verifying that the occurrence process, as modelled here, is in fact ergodic.

## 9   Integrated results

Results for the integrated distribution at one of the Italian stations are shown in Figure 2. By integrating out the random shocks we are left with a smooth representation of time dependency alone. The annual cycles are clear. Trends are also evident over the 30 year period. The mean rainfall amount is decreasing significantly, although the probability of rain on a given day of the year remains much the same. Rain is occurring no less frequently, but it is occurring in smaller amounts. Note also that the winter rainfall (the upper envelope of the mean) is decreasing more rapidly than the summer rainfall (the lower envelope of the mean) so that the difference between the two is narrowing.

## 10   Conclusions

This paper provides a new example of time series modelling using neural networks. The use of a mixture of a discrete distribution and a gamma distribution emphasises the general principle that the "error function" for a neural network depends on the particular statistical model used for the target data. The use of cyclic variables $\sin \tau$ and $\cos \tau$ as inputs shows how the problem of selecting the number of harmonics required for a Fourier series analysis of seasonality can be solved adaptively. Long term trends can also be modelled by the use of a linear time variable, although both this and the last feature require the presence of a suitable regularizer to avoid overfitting. Lastly we have seen how a suitable form of integration can be used to extract the underlying cycles and trends from noisy data. These techniques can be adapted to the analysis of time series drawn from other domains.

## Acknowledgement

I am indebted to Professor Helen Rendell of the School of Chemistry, Physics and Environmental Sciences, University of Sussex, for kindly supplying the rainfall data and for valuable discussions.

## Footnotes

[1]It would be straightforward to use a mixture of gammas, or exponentials, with time-dependent mixture components. A single gamma was chosen for simplicity to illustrate the approach.

[2]A similar approach to modelling conditional distributions, by having the network output distributional parameters, is used, for example, by Ghahramani & Jordan [6], Nix & Weigend [10], Bishop & Legleye [3], Williams [14], Baldi & Chauvin [2].

[3]Note that both $\sin n\tau$ and $\cos n\tau$ can be expressed as non-linear functions of $\sin \tau$ and $\cos \tau$, which can be approximated by the network.

[4]The English stations were at Cromptons, Falmer, Kemsing, Petworth, Rotherfield; the Italian stations were at Monte Oliveto, Pisticci, Pomarico, Siena, Taverno d'Arbia.

[5]In view of the number of lags used as inputs, the conditional probability would necessarily be constant after 10 days apart from seasonal effects. In fact this is the last quarter of 1951 and the incidence of rain is increasing here at that time of year.

## References

[1] D. E. Amos. A portable fortran subroutine for derivatives of the psi function. *ACM Transactions on Mathematical Software*, 9:494–502, 1983.

[2] P. Baldi and Y. Chauvin. Hybrid modeling, HMM/NN architectures, and protein applications. *Neural Computation*, 8:1541–1565, 1996.

[3] C. M. Bishop and C. Legleye. Estimating conditional probability densities for periodic variables. In G. Tesauro, D. Touretzky, and T. Leen, editors, *Advances in Neural Information Processing Systems 7*, pages 641–648. The MIT Press, 1995.

[4] E. H. Chin. Modelling daily precipitation occurrence process with Markov chain. *Water Resources Research*, 13:949–956, 1977.

[5] P. Gates and H. Tong. On Markov chain modelling to some weather data. *Journal of Applied Meteorology*, 15:1145–1151, 1976.

[6] Z. Ghahramani and M. I. Jordan. Supervised learning from incomplete data via an EM approach. In Jack D. Cowan, Gerald Tesauro, and Joshua Alspector, editors, *Advances in Neural Information Processing Systems 6*, pages 120–127. Morgan Kaufmann, 1994.

[7] N. T. Ison, A. M. Feyerherm, and L. D. Bark. Wet period precipitation and the gamma distribution. *Journal of Applied Meteorology*, 10:658–665, 1971.

[8] R. W. Katz. Precipitation as a chain-dependent process. *Journal of Applied Meteorology*, 16:671–676, 1977.

[9] R. W. Katz and M. B. Parlange. Effects of an index of atmospheric circulation on stochastic properties of precipitation. *Water Resources Research*, 29:2335–2344, 1993.

[10] D. A. Nix and A. S. Weigend. Learning local error bars for nonlinear regression. In Gerald Tesauro, David S. Touretzky, and Todd K. Leen, editors, *Advances in Neural Information Processing Systems 7*, pages 489–496. MIT Press, 1995.

[11] W. H. Press, B. P. Flannery, S. A. Teukolsky, and W. T. Vetterling. *Numerical Recipes in C*. Cambridge University Press, 2nd edition, 1992.

[12] R. D. Stern and R. Coe. A model fitting analysis of daily rainfall data, with discussion. *Journal of the Royal Statistical Society A*, 147(Part 1):1–34, 1984.

[13] P. M. Williams. Bayesian regularization and pruning using a Laplace prior. *Neural Computation*, 7:117–143, 1995.

[14] P. M. Williams. Using neural networks to model conditional multivariate densities. *Neural Computation*, 8:843–854, 1996.

[15] D. A. Woolhiser. Modelling daily precipitation—progress and problems. In Andrew T. Walden and Peter Guttorp, editors, *Statistics in the Environmental and Earth Sciences*, chapter 5, pages 71–89. Edward Arnold, 1992.

[16] D. A. Woolhiser and G. G. S. Pegram. Maximum likelihood estimation of Fourier coefficients to describe seasonal variation of parameters in stochastic daily precipitation models. *Journal of Applied Meteorology*, 18:34–42, 1979.